# Spectral learning of linear dynamics from generalised-linear observations with application to neural population data

**Lars Buesing**[*], **Jakob H. Macke**[*,†] , **Maneesh Sahani**

Gatsby Computational Neuroscience Unit
University College London, London, UK
`{lars, jakob, maneesh}@gatsby.ucl.ac.uk`

## Abstract

Latent linear dynamical systems with generalised-linear observation models arise in a variety of applications, for instance when modelling the spiking activity of populations of neurons. Here, we show how spectral learning methods (usually called subspace identification in this context) for linear systems with linear-Gaussian observations can be extended to estimate the parameters of a generalised-linear dynamical system model despite a non-linear and non-Gaussian observation process. We use this approach to obtain estimates of parameters for a dynamical model of neural population data, where the observed spike-counts are Poisson-distributed with log-rates determined by the latent dynamical process, possibly driven by external inputs. We show that the extended subspace identification algorithm is consistent and accurately recovers the correct parameters on large simulated data sets with a single calculation, avoiding the costly iterative computation of approximate expectation-maximisation (EM). Even on smaller data sets, it provides an effective initialisation for EM, avoiding local optima and speeding convergence. These benefits are shown to extend to real neural data.

## 1 Introduction

Latent linear dynamical system (LDS) models, also known as Kalman-filter models or linear-Gaussian state-space models, provide an important framework for modelling shared temporal structure in multivariate time series. If the observation process is linear with additive Gaussian noise, then there are many established options for parameter learning. Inference of the dynamical state in such a model can be performed exactly by Kalman smoothing [1] and so the expectation-maximisation (EM) algorithm may be used to find a local maximum of the likelihood [2]. An alternative is the spectral approach known as subspace identification (SSID) in the engineering literature [3, 4, 5]. This is a method-of-moments-based estimation process, which, like other spectral methods, provides estimators that are non-iterative, consistent and do not suffer from the problems of multiple optima that dog maximum-likelihood (ML) learning in practice. However, they are not as statistically efficient as the true (global) ML estimator. Thus, a combined approach often produces the best results, with the SSID-based parameter estimates being used to initialise the EM iterations.

Many real-world data sets, however, are not well described by a linear-Gaussian output process. Of particular interest to us here are multiple neural spike-trains measured simultaneously by arrays of electrodes [6, 7], which are best treated either as multivariate point-processes or, after binning, as a time series of vectors of small integers. In either case the event rates must be positive, precluding a linear mapping from the Gaussian latent process, and the noise distribution cannot accurately be

---

[*] These authors contributed equally. [†] Current Affiliation: Max Planck Institute for Biological Cybernetics and Bernstein Center for Computational Neuroscience Tübingen

modelled as normal. Similar point-process or count data may arise in many other settings, such as seismology or text modelling. More generally, we are interested in the broad class of *generalised-linear* output models (defined by analogy to the generalised-linear regression model [8]), where the expected value of an observation is given by a monotonic function of the latent Gaussian process, with an arbitrary (most frequently exponential-family) distribution of observations about this mean.

For such models exact inference, and therefore exact EM, is not possible. Instead, approximate ML learning relies on either Monte-Carlo or deterministic approximations to the posterior. Such methods may be computationally intensive, suffer from varying degrees of approximation error, and are subject to the same concerns about multiple likelihood optima as is the linear-Gaussian case[2] Thus, a consistent spectral method is likely to be of particular value for such models. In this paper we show how the SSID approach may be extended to yield consistent estimators for generalised-linear-output LDS (gl-LDS) models. In experiments with simulated and real neural data, we show that these estimators may be better than those provided by approximate EM when given sufficient data. Even when data are few, the approach provides a valuable initialisation to approximate EM.

## 2   Theory

We briefly review the Ho-Kalman SSID algorithm [10] for linear-Gaussian LDS models, before extending it to the gl-LDS case. Using this framework, we derive and then evaluate an algorithm to fit models of Poisson-distributed count data with log-rates generated by an LDS.

### 2.1   SSID for LDS models with linear-Gaussian observations

Let $q$-dimensional observations $\mathbf{y}_t$, $t \in \{1, \ldots, T\}$ depend on a $p$-dimensional latent state $\mathbf{x}_t$, described by a linear first-order auto-regressive process with Gaussian initial distribution and Gaussian innovations:

$$\begin{aligned} \mathbf{x}_1 &\sim \mathcal{N}(\mathbf{x}_0, Q_0) \\ \mathbf{x}_{t+1} \mid \mathbf{x}_t &\sim \mathcal{N}(A\mathbf{x}_t, Q) \\ \mathbf{z}_t &= C\mathbf{x}_t + \mathbf{d} \\ \mathbf{y}_t \mid \mathbf{z}_t &\sim \mathcal{N}(\mathbf{z}_t, R). \end{aligned} \tag{1}$$

Here, $\mathbf{x}_0$ and $Q_0$ are the mean and covariance of the initial state and $Q$ is the covariance of the innovations. The dynamics matrix $A$ models the temporal dependence of the process $\mathbf{x}$. The variable $\mathbf{z}_t$ of dimension $q$ is defined as an affine function of the latent state $\mathbf{x}_t$, parametrised by the loading matrix $C$ and the mean parameter $\mathbf{d}$. Given $\mathbf{z}_t$, observations are independently distributed around this value with covariance $R$. Furthermore let $\Pi := \lim_{t \to \infty} \text{Cov}[\mathbf{x}_t]$ denote the covariance of the stationary marginal distribution if the system is stable (i.e. if the spectral radius of $A$ is $< 1$).

Provided the generative model is stationary (i.e., $\mathbf{x}_0 = 0$ and $Q_0 = \Pi$), SSID algorithms yield consistent estimates of the parameters $A, C, Q, R, \mathbf{d}$ without iteration. We adopt an approach to SSID based on the Ho-Kalman method [10, 4]. This algorithm takes as input the empirical estimate of the so-called "future-past Hankel matrix" $H$ which is defined as the cross-covariance between time-lagged vectors $\mathbf{y}_t^+$ (the "future") and $\mathbf{y}_t^-$ (the "past") of the observed data:

$$H := \text{Cov}[\mathbf{y}_t^+, \mathbf{y}_t^-] \qquad \mathbf{y}_t^+ := \begin{pmatrix} \mathbf{y}_t \\ \vdots \\ \mathbf{y}_{t+k-1} \end{pmatrix} \qquad \mathbf{y}_t^- := \begin{pmatrix} \mathbf{y}_{t-1} \\ \vdots \\ \mathbf{y}_{t-k} \end{pmatrix}.$$

The parameter $k$ is called the Hankel size and has to be chosen so that $k \geq p$. The key to SSID is that $H$ (which is independent of $t$ as stationarity is assumed) has rank equal to the dimensionality $p$ of the linear dynamical state. Indeed, it is straightforward to show that the Hankel matrix can be decomposed in terms of the model parameters $A, C, \Pi$,

$$H = [C^\top \ (CA)^\top \ \ldots \ (CA^{k-1})^\top]^\top \cdot [A\Pi C^\top \ A^2\Pi C^\top \ \ldots \ A^k \Pi C^\top]. \tag{2}$$

The SSID algorithm first takes the singular value decomposition (SVD) of the empirical estimate $\widehat{H}$ of $H$ to recover a two-part factorisation as in (2) given a user-defined latent dimensionality $p$ (a suitable $p$ may be estimated by inspection of the singular value spectrum of $\hat{H}$). From this low-rank

approximation to $\widehat{H}$ the model parameters $A$, $C$ as well as the covariances $Q$ and $R$ can be found by linear regression and by solving an algebraic Riccati equation; $\mathbf{d}$ is given simply by the empirical mean of the data. However, this specific procedure works only for linear systems with Gaussian observations and innovations, and not for models which feature non-linear transformations or non-Gaussian observation models. Indeed, we find that linear SSID methods can yield poor results when applied directly to count-process data. Although SSID techniques have been developed for observations that are Gaussian-distributed around a mean that is a nonlinear function of the latent state [5], we are unaware of SSID methods that address arbitrary observation models.

## 2.2 SSID for gl-LDS models by moment conversion

Consider now the gl-LDS in which the Gaussian observation process of model (1) is replaced by the following more general observation model. We assume $y_{t,i} \perp y_{t,j} \mid \mathbf{z}_t$; i.e. observation dimensions are independent given $\mathbf{z}_t$. Further, let $y_{t,i} \mid \mathbf{z}_t$ be arbitrarily distributed around a (known) monotonic element-wise nonlinear mapping $f(\cdot)$ such that $\mathbb{E}[\mathbf{y}_t|\mathbf{z}_t] = f(\mathbf{z}_t)$. Following the theory of generalised linear modelling, we also assume that the variance of the observation distribution is a (known) function $V(\cdot)$ of its mean.[3]

Our extension to SSID for such models is based on the following idea. The variables $\mathbf{z}_1, \ldots, \mathbf{z}_T$ are jointly normal, so in principle we can apply standard SSID algorithms to $\mathbf{z}$. Although $\mathbf{z}$ is unobserved, we can use the fact that the observation model dictates a computable relationship between the moments of $\mathbf{y}$ and those of $\mathbf{z}$. This allows us to determine the future-past Hankel matrix of $\mathbf{z}$ from the moments of $\mathbf{y}$, which can then be fed into standard SSID algorithms. Consider the covariance matrix $\mathrm{Cov}[\mathbf{y}^{\pm}]$ of the combined $2kq$-dimensional future-past vector $\mathbf{y}^{\pm}$ which is defined by stacking $\mathbf{y}^{+}$ and $\mathbf{y}^{-}$ (here and henceforth we drop the subscripts $t$ as unnecessary given the assumed stationarity of the process). Denote the mean and covariance matrix of the normal distribution of $\mathbf{z}^{\pm}$ (defined analogously to $\mathbf{y}^{\pm}$) by $\boldsymbol{\mu}$ and $\Sigma$. We then have,

$$\mathbb{E}[y_i^{\pm}] = \mathbb{E}_z[f(z_i^{\pm})] \quad =: \quad \alpha(\mu_i, \Sigma_{ii}) \qquad (3)$$

$$\mathbb{E}[(y_i^{\pm})^2] = \mathbb{E}_z[\mathbb{E}_{y|z}[(y_i^{\pm})^2]] = \mathbb{E}_z[f(z_i^{\pm})^2 + V(f(z_i^{\pm}))] \quad =: \quad \beta(\mu_i, \Sigma_{ii}). \qquad (4)$$

The functions $\alpha(\cdot)$ and $\beta(\cdot)$ are given by Gaussian integrals with mean $\mu_i$ and variance $\Sigma_{ii}$ over the functions $f(\cdot)$ and $f^2(\cdot) + V(f(\cdot))$, respectively. For off-diagonal second moments we have ($i \neq j$):

$$\mathbb{E}[y_i^{\pm} y_j^{\pm}] = \mathbb{E}_z[\mathbb{E}_{y|z}[y_i^{\pm}] \cdot \mathbb{E}_{y|z}[y_j^{\pm}]] = \mathbb{E}_z[f(z_i^{\pm})f(z_j^{\pm})] \quad =: \quad \gamma(\mu_i, \Sigma_{ii}, \mu_j, \Sigma_{jj}, \Sigma_{ij}). \qquad (5)$$

Equations (3)-(5) are a $4kq + kq(2kq - 1)$ system of non-linear equations in $4kq + kq(2kq - 1)$ unknowns $\boldsymbol{\mu}$, $\Sigma$ (with symmetric $\Sigma = \Sigma^{\top}$). The equations above can be solved efficiently by separately solving one 2-dimensional system (equations 3-4) for each pair of unknowns $\mu_i$, $\Sigma_{ii}$, $\forall i \in \{1, \ldots, kq\}$. Once the $\mu_i$ and $\Sigma_{ii}$ are known, equation (5) reduces to a 1-dimensional nonlinear equation for $\Sigma_{ij}$ for each pair of indices ($i < j$). The upper-right block of the covariance matrix $\Sigma$ then provides an estimate of the future-past Hankel matrix $\mathrm{Cov}[\mathbf{z}^{+}, \mathbf{z}^{-}]$ which can be decomposed as in standard Ho-Kalman SSID.

## 2.3 SSID for Poisson dynamical systems (PLDSID)

We now consider in greater detail a special case of the gl-LDS model, which is of particular interest in neuroscience applications. The observations in this model are (when conditioned on the latent state) Poisson-distributed with a mean that is exponential in the output of the dynamical system,

$$y_{t,i} \mid z_{t,i} \quad \sim \quad \mathrm{Poisson}[\exp(z_{t,i})].$$

We call this model, which is a special case of a Log-Gaussian Cox Process [11], a Poisson Linear Dynamical System (PLDS). PLDS and close variants have recently been applied for modelling multi-electrode recordings [12, 13, 14, 15]. In these applications, $y_{t,i}$ models the spike-count of neuron $i$ in time-bin $t$ and its log-firing-rate (which we will refer to as the "input to neuron $i$") is given by $z_{t,i}$. Estimation of the model parameters $\Theta = (A, C, Q, \mathbf{x}_0, Q_0, \mathbf{d})$ often depends on approximate likelihood maximisation, using EM with an approximate E-step [16, 9]. The exponential nonlinearity ensures that the posterior distribution $p(\mathbf{x}_{1\ldots T}|\mathbf{y}_{1\ldots T}, \Theta)$ is a log-concave function of $\mathbf{x}_{1\ldots T}$ [17], making its mode easy to find and justifying unimodal approximations (such as that of Laplace). However, the typical data likelihood is nonetheless multimodal and the approximations may introduce bias in estimation [18].

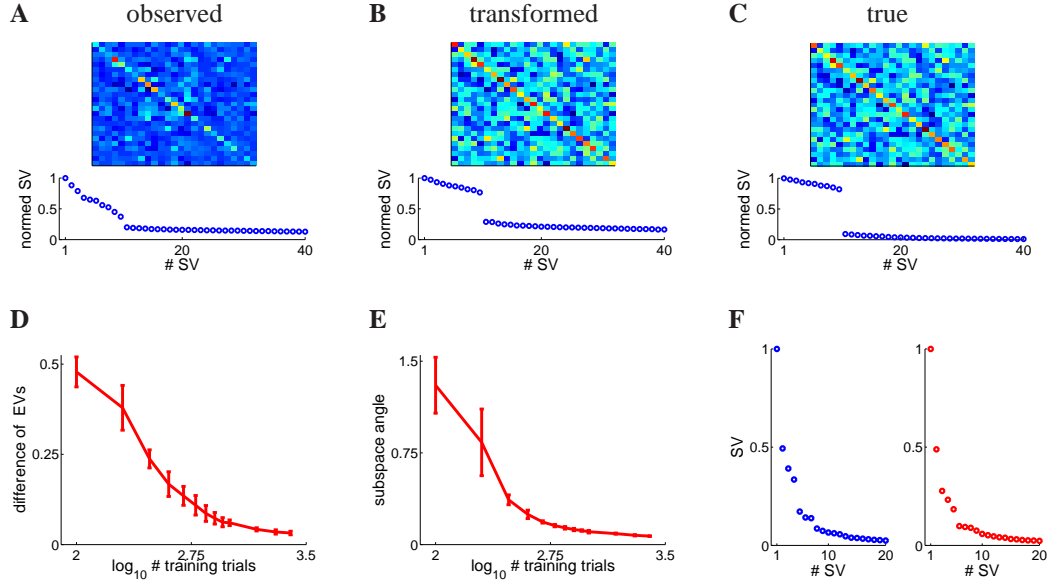

Figure 1: **Moment conversion uncovers low-rank structure in artificial data. A)** Time-lagged covariance matrix $\mathrm{Cov}[\mathbf{y}_{t+1}, \mathbf{y}_t]$ and the singular value (SV) spectrum of the full Hankel matrix $H = \mathrm{Cov}[\mathbf{y}^+, \mathbf{y}^-]$ computed from the observed count data (artificial data set I). The spectrum decays gradually. **B)** Same as A) but after moment conversion. The transformed Hankel matrix now exhibits a clear cut-off in the spectrum, indicative of low underlying rank. **C)** Same as A) and B) but computed from the (ground truth) log-rates $\mathbf{z}$, illustrating the true low-rank structure in the data. **D)** Summed absolute difference of the eigenvalue spectra of the ground truth dynamics matrix $A$ and the one identified by PLDSID. The difference decreases with increasing data set size, indicating that PLDSID estimates are consistent. **E)** Same as C) but for the angle between the subspaces spanned by the loading matrix of the ground truth and estimated models. **F)** SV spectrum of the Hankel matrix of multi-electrode data before (left) and after (right) moment conversion.

Under the PLDS model, the equations (3)-(5) can be solved analytically (see also [19] and the supplementary material for details),

$$\mu_i = 2\log(m_i) - \frac{1}{2}\log(S_{ii} + m_i^2 - m_i) \tag{6}$$

$$\Sigma_{ii} = \log(S_{ii} + m_i^2 - m_i) - \log(m_i^2) \tag{7}$$

$$\Sigma_{ij} = \log(S_{ij} + m_i m_j) - \log(m_i m_j), \tag{8}$$

where $m_i$ and $S_{ij}$ denote the empirical estimates of $\mathbb{E}[y_i^\pm]$ and $\mathrm{Cov}[y_i^\pm, y_j^\pm]$, respectively. One can see that the above equations do not have solutions if any one of the terms in the logarithms is non-positive, which may happen with finitely sampled moments or a misspecified model. We therefore scale the matrix $S$ (by left and right multiplication with the same diagonal matrix) such that all Fano factors that are initially smaller than 1 are set to a given threshold (in simulations we used $1 + 10^{-2}$). This procedure ensures that there exists a unique solution $(\boldsymbol{\mu}, \Sigma)$ to the moment conversion (6)-(8). It is still the case that the resulting matrix $\Sigma$ might not be positive semidefinite [20], but this can be rectified by finding its eigendecomposition, thresholding the eigenvalues (EVs) and then reconstructing $\Sigma$.

For sufficiently large data sets generated from a "true" PLDS model, observed Fano factors will be greater than one with high probability. In such cases, the moment conversion asymptotically yields the unique correct moments $\boldsymbol{\mu}$ and $\Sigma$ of the Gaussian log-rates $\mathbf{z}$. Assuming stationarity, the Ho-Kalman SSID yields consistent estimates of $A, C, Q, \mathbf{d}$ given the true $\boldsymbol{\mu}$ and $\Sigma$. Hence, the proposed two-stage method yields consistent estimates of the parameters $A, C, Q, \mathbf{d}$ of a stationary PLDS. In the remainder, we call this algorithm PLDSID.

It is often of interest to model the conditional distribution of the observables $\mathbf{y}$ given some external, observed covariate or "input" $\mathbf{u}$. In neuroscience, for instance, $\mathbf{u}$ might be a sensory stimulus influencing retinal [14] or other sensory spiking activity. Fortunately, provided that the external inputs are Gaussian-distributed and perturb the dynamics linearly, PLDSID can be extended to identify the

parameters of this augmented model. Let $\mathbf{u}_t$ denote the $r$-dimensional observed external input at time $t$, and assume that $\mathbf{u}_1, \ldots, \mathbf{u}_T$ are jointly normal and influence the latent state of the dynamical process linearly and instantaneously (through a $p \times r$ matrix $B$):

$$\mathbf{x}_{t+1} \mid \mathbf{x}_t, \mathbf{u}_t \quad \sim \quad \mathcal{N}(A\mathbf{x}_t + B\mathbf{u}_t, Q),$$

The dynamical state $\mathbf{x}_t$ is then observed through a generalised-linear process as before, and we define future-past vectors for all relevant time series. In this case, the N4SID algorithm [3] can perform subspace identification based on the joint covariance of $\mathbf{u}^{\pm}$ and $\mathbf{z}^{\pm}$. Although this covariance is not observed directly in the gl-LDS case, our assumptions make $\mathbf{u}^{\pm}$ and $\mathbf{z}^{\pm}$ jointly normal and so we can use moment transformation again to estimate the required covariance from the observed covariance of $\mathbf{u}^{\pm}$ and $\mathbf{y}^{\pm}$. For the Poisson model with exponential nonlinearity, this transformation remains closed-form, and in combination with N4SID yields consistent estimates of the PLDS parameters and the input-coupling matrix $B$. [4] Further details are provided in the supplementary material.

# 3 Results

We investigated the properties of the proposed PLDSID algorithm in numerical experiments, using both artificial data and multi-electrode recordings of neural activity.

## 3.1 PLDSID infers the correct parameters given sufficiently large synthetic data sets

We used three artificial data sets to evaluate our algorithm, each consisting of 200 time-series ("trials"), with each trial being of length $T = 100$ time steps. Time-series were generated by sampling from a stationary ground truth PLDS with $p = 10$ latent and $q = 25$ observed dimensions. Count averages across time-bins and neurons ranged from 0.15 to 0.2, corresponding to 15–20 Hz if the time-step size $dt$ is taken to be 10 ms (the binning used for the multi-electrode recordings, see below). The dynamics matrices $A$ had eigenvalues corresponding to auto-correlation time constants ranging from < 1 time step (data set III), through $3\,dt$ (data set I) to $20\,dt$ (data set II). The loading matrices $C$ were generated from a matrix with orthonormal columns and by a subsequent scaling with 12.5 (data set I) or 5 (data sets II and III). This resulted in instantaneous correlations that were comparable to (average absolute correlation coefficient data set I: $\bar{c} = 2 \cdot 10^{-2}$) or smaller than (data sets II, III: $\bar{c} = 3.5 \cdot 10^{-3}$) those observed in the cortical multi-electrode recordings used below ($\bar{c} = 2.2 \cdot 10^{-2}$). Hence, all our artificial data sets either roughly match (data sets I, II) or substantially underestimate (data set III) the correlation-structure of typical cortical multi-cell recordings. Additionally, we generated a data set for identifying PLDS models with external input by driving the ground truth PLDS of data set II with a 3 dimensional Gaussian AR(1) process $\mathbf{u}_t$; the coupling matrix $B$ was generated such that $B\mathbf{u}_t$ had the same covariance as the innovations $Q$. A Hankel size $k = 10$ was used for all experiments with artificial data.

We first illustrate the moment conversion defined by equations (6)-(8) on artificial data set I. Fig. 1A shows the time-lagged cross-covariance $\mathrm{Cov}[\mathbf{y}_{t+1}, \mathbf{y}_t]$ as well as the singular value (SV) spectrum of the full future-past Hankel matrix $H = \mathrm{Cov}[\mathbf{y}^+, \mathbf{y}^-]$ (normalised such that the largest SV is 1), both estimated from 200 trials, with a Hankel size of $k = 10$. The raw spectrum gradually decays towards small values but does not show a clear low-rank structure of the future-past Hankel matrix $H$. In contrast, Fig. 1B shows the output of the moment transformation yielding an approximation of the cross-covariance $\mathrm{Cov}[\mathbf{z}_{t+1}, \mathbf{z}_t]$ of the underlying inputs. Further the SV spectrum of the full, transformed future-past Hankel matrix $\mathrm{Cov}[\mathbf{z}^+, \mathbf{z}^-]$ is shown. The latter is dominated by only a few SVs, whose number matches the dimension of the ground truth system $p = 10$, clearly indicating a low-rank structure. On this synthetic data set, we also have access to the underlying inputs. One can see that the transformed Hankel matrix Fig. 1B as well as its SV spectrum are close to the ones computed from the underlying inputs shown in Fig. 1C.

We also evaluated the accuracy of the parameters identified by PLDSID as a function of the training set size. Fig. 1D shows the difference between the spectra (i.e., the summed absolute differences between sorted eigenvalues) of the identified and the ground truth dynamics matrix $A$. The spectrum of $A$ is an important characteristic of the model, as it determines the time-constants of the underlying dynamics. It can be seen that the difference between the spectra decreases with increasing data set size (Fig. 1D), indicating that our method asymptotically identifies the correct dynamics. Furthermore, Fig. 1E shows the subspace-angle between the true loading matrix $C$ and the one estimated

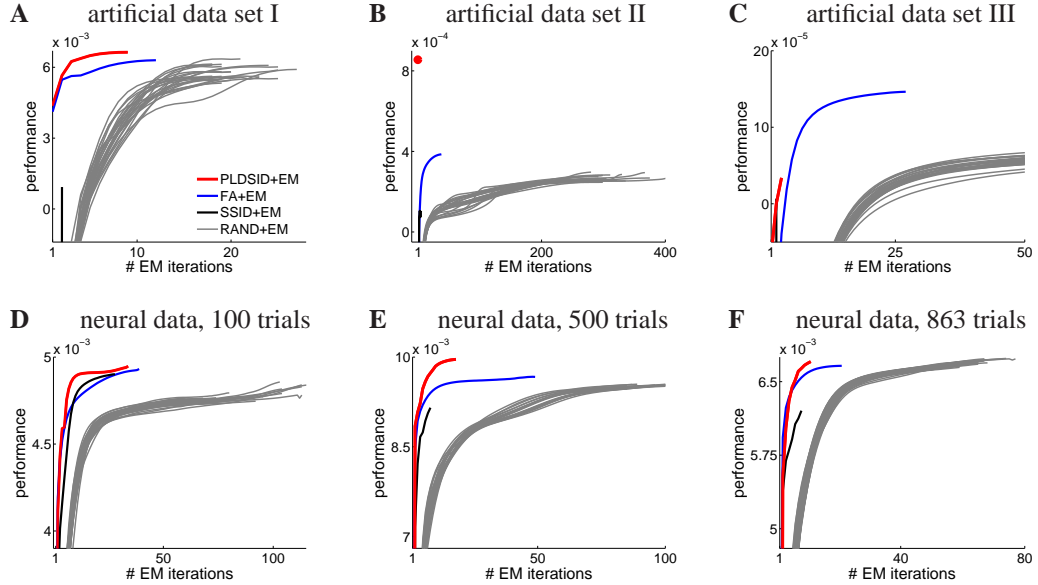

Figure 2: **PLDSID is a good initialiser for EM.** Cosmoothing performance on the training set as a function of the number of EM iterations for different initialisers on various data sets. **A)** Artificial data set consisting of 200 trials and 25 observed dimensions. EM initialised by PLDSID converges faster and achieves higher training performance than EM initialised with FA, Gaussian SSID or random parameter values. **B)** Same as A) but for data with lower instantaneous correlations and longer auto-correlation. EM does not improve the performance of PLDSID on this data set. **C)** Same as A) but for data with negligible temporal correlations and low instantaneous correlations. For this weakly structured data set, PLDSID-EM does not work well. **D)** 100 trials of multi-electrode recordings with 86 observed dimensions (spike-sorted units). **E)** Same as D) but of data set size 500 trials, and only using the 40 most active units **F)** Same as D) but for 863 trials with all 86 units.

by PLDS. As for the dynamics spectrum, the identified loading matrix approaches the true one for increasing training set size.

Next, we investigated the usefulness of PLDSID as an initialiser for EM. We compared it to 3 different methods, namely initialisation with random parameters (with 20-50 restarts), factor analysis (FA) and Gaussian SSID. The quality of these initialisers was assessed by monitoring performance of the identified parameters as a function of EM iterations after initialisation. Good initial parameter values yield fast convergence of EM in few iterations to high performance values, whereas poor initialisations are characterised by slow convergence and termination of EM in poor local maxima (or, potentially, shallow regions of the likelihood). Fast convergence of EM is an important issue when dealing with large data sets, as EM iterations become computationally expensive (see below). We monitor performance by evaluating the so-called cosmoothing performance on the training data, a measure for cross-prediction performance described elsewhere in detail [21, 15]. This measure yielded more reliable and robust results than computing the likelihood, as the latter cannot be computed exactly and approximations can be numerically unreliable. We evaluated performance on the training set, as we were interested in comparing fitting-performance of the algorithms for the same model, and not the generalisation error of the model itself.

Fig. 2A to C show the results of this comparison on three different artificial data sets. On data set I (Fig. 2A), which was designed to have short auto-correlation time constants but pronounced instantaneous correlations between the observed dimensions, PLDSID initialisation leads to superior performance compared to competing methods. For the same number of EM iterations (which is a good proxy of invested computation time, see below), it resulted in better co-smoothing performance. Furthermore, the PLDSID+EM parameters converge to a better local optimum than those initialised by the other methods. Hence, on this data set, our initialisation yields both faster computation time and better final results. The second artificial data set featured smaller instantaneous correlations between dimensions but longer auto-correlation time constants. As can be seen in Fig. 2B, the PLDSID initialisation here yields parameters which are not further improved by EM iterations whereas EM with other initialisations becomes stuck in poor local solutions.

By contrast, we found PLDSID not to yield useful parameter values on data sets which do not have temporal correlations (Fig. 2C), and only very small instantaneous correlations across neurons (average instantaneous absolute-correlation $\bar{c} = 3.5 \cdot 10^{-3}$). For this particular data set, PLDSID and Gaussian SSID both yielded poor parameters compared to factor analysis. In general, we observed that PLDSID compares favourably to the other initialisation methods on any data sets we investigated as long as it exhibits shared variability across dimensions and time, and it was observed to work particularly well when correlations were substantial. Fig. 3 shows results for identification of a PLDS model driven by external inputs. The proposed PLDSID method identifies better PLDS parameters, including the coupling matrix $B$, than alternative methods. Notably, identifying the parameters with the PLDSID-variant that ignores external input (and setting the initial value $B = 0$ for EM) clearly results in suboptimal parameters.

## 3.2 Expectation Maximisation initialised by PLDSID identifies better models on neural data

We move now to examine the value of PLDSID in providing initial parameter values for subsequent EM iterations on multi-electrode recordings of neural activity. Such data sets are challenging for statistical modelling as they are high-dimensional (on the order of $10^2$ observed dimensions), sparse (on the order of $10$ Hz of spiking activity) and show shared variability across time and dimensions. The experimental setup, acquisition and preprocessing of the data set are documented in detail elsewhere [22]. Briefly, spiking activity was acquired from a 96-channel silicon electrode array (Blackrock, Salt Lake City, UT) implanted in motor areas of the cortex of a rhesus macaque performing a delayed center-out reach task. For the analysis presented in this paper, we used data from a single recording session consisting in total of 863 trials, each truncated to be of length $1$ s with 86 distinct single and multi-units identified by spike sorting. The data had an average firing rate of $10.7$ Hz and it was binned at $10$ ms which resulted in $9.9\%$ of bins containing at least one spike.

First, we investigated the SV spectrum of the future-past Hankel matrix computed either from the count-observations of the data, or from the inferred underlying inputs (using Hankel size $k = 30$ and all trials, see Fig. 1F). While we did not observe a marked difference between the two spectra, both spectra indicate that the data can be well described using a small number of singular values. Based on these spectra, we used a dimensionality of $q = 10$ for subsequent simulations.

Next, we compared PLDSID to FA and Gaussian SSID initialisations for EM on two different subsets as well as the whole multi-electrode recording data set. Fig. 2D shows the performance of EM with the different initialisations using a training set of modest size (100 trials, Hankel size $k = 10$). PLDSID provides the most appropriate initialisation for EM, allowing it to converge rapidly to better parameter values than are found starting from either the FA or SSID estimates. This effect was still more pronounced for a larger training set of 500 trials, but including only the 40 most active neurons from the original data (Fig. 2E, Hankel size $k = 30$). We also applied all of the methods to the complete data set consisting of 863 trials with all 86 observed neurons (Hankel size $k = 30$). The results plotted in Fig. 2F indicate that again PLDSID provided the most useful initialisation for EM. Interestingly, on this data set EM with random initialisations eventually identifies parameters with performances comparable to PLDSID+EM. However, random initialisation leads to slow convergence and thus requires substantial computation, as described below. Gaussian SSID yielded poor values for parameters on all data sets, leading EM to terminate in poor local optima after only a few iterations. We note that, because of the use of the Laplace approximation during inference (as well as our non-likelihood performance measure) EM is not guaranteed to increase performance at each iteration, and, in practice, sometimes terminated after rather few iterations.

## 3.3 PLDSID improves training time by orders of magnitude compared to conventional EM

The computational time needed to identify PLDS parameters might prove to be an important issue in practice. For example, when using a PLDS model as part of an algorithm for brain-machine interfacing [12], the parameters must be identified during an experimental session. For multi-electrode recording data of commonly-encountered size, and using our implementation of EM, inference of parameters under these time-constraints would be infeasible. Thus, an ideal parameter initialisation method will not only improve the robustness of the identified parameters, but also reduce the computational time needed for EM convergence. Clearly, the computer time needed will depend on the implementation, hardware and the properties and size of the data used. We used an EM-algorithm with a global-Laplace approximation in the E-step [23, 15], and a conjugate-gradient-based optimi-

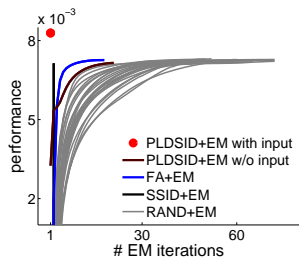

Figure 3: **Identification of PLDS models with external inputs.** Same as Fig. 2 B) but for an artificial data set which is generated by sampling from a PLDS with external input. Using the variant of PLDSID which also identifies the coupling matrix $B$ yields yields the best parameters. In contrast, using the PLDSID variant which does not estimate $B$ ($B$ is initialised at $0$) yields parameters which are of the same quality as alternative methods.

sation method in the M-step implemented in Matlab. Alternative methods based on variational approximations or MCMC-sampling have been reported to be more costly than Laplace-EM [13, 24].

For all of the data sets used above, one single EM iteration in our implementation was substantially more costly than parameter initialisation by PLDSID (Fig. 2D: factor $6.4$, Fig. 2E: factor $4.0$, Fig. 2F: factor $1.4$). In addition, EM started with random initialisation still yielded worse performance than with PLDSID initialisation even after 50 iterations (see Figure 2). Thus, even with a conservative estimate, PLDSID initialisation reduces computational cost by at least a factor of $50$ compared to random initialisation. Both PLDSID and EM have a time computational complexity which is proportional to the size $NT$ of the data set (where $N$ is the number of trials and $T$ is the trial length). However, in PLDSID, only the cost $O(NTpq^2)$ of calculating the Hankel-matrix scales with the data set size (assuming $k$ is of order $p$). This simple covariance calculation was much cheaper in our experiments than the moment conversion with cost $O(pq^2)$ or the SVD with cost $O(p^3q^3)$, both of which are independent of the data set size $NT$. In contrast, each iteration of EM requires at least $O(NT(p^3 + pq))$ time. Therefore, the computational advantage of PLDSID is expected to be especially great for large data sets. This is also the regime where the performance benefit is most pronounced.

# 4 Discussion

We investigated parameter estimation for linear-Gaussian state-space models with *generalised-linear* observations and presented a method for parameter identification in such models which builds on the extensive subspace-identification literature for fully Gaussian state-space models. In numerical experiments we studied a special case of the proposed algorithm (PLDSID) for linear state-space models with conditionally Poisson-distributed observations. We showed that PLDSID yields consistent estimates of the model parameters without requiring iterative computation. Although this method generally makes less efficient use of available training data than do maximum likelihood methods, we found that it sometimes outperformed likelihood hill-climbing by EM from random initial conditions in practice (presumably due to optimisation difficulties). Even when this was not the case, EM initialised with the results of PLDSID converged in fewer iterations, and to a better parameter estimate than when it was initialised randomly, or by other methods—an effect seen with multiple artificial and multi-electrode recording data sets. As the practical computational difficulties of parameter estimation (slow convergence and shallow optima in parameter estimation with EM) in this model are substantial, our algorithm facilitates the use of linear state-space models with non-Gaussian observations in practice.

While proven here in the Poisson case, the underlying moment-transformation algorithm is flexible and can be applied to a wide range of gl-LDS models. Of particular interest for neural data might be a dynamical system model which precisely reproduced the marginal distribution of integer observations for each observed dimension (by using a 'Discretised Gaussian' [20] as the observation model). By contrast, the need for tractability in sampling or deterministic approximations for inference often limits the range of models in which EM is practical.

**Acknowledgements** Supported by the Gatsby Charitable Foundation; an EU Marie Curie Fellowship to JHM (hosted by MS); DARPA REPAIR N66001-10-C-2010 and NIH CRCNS R01-NS054283 to MS; as well as the Bernstein Center Tübingen funded by the German Ministry of Education and Research (BMBF; FKZ: 01GQ1002). We would like to thank Krishna V. Shenoy and members of his laboratory for many useful discussions as well as for generously sharing their data with us.

## Footnotes

[2]A recent paper [9] has argued that the log-likelihood of a model with Poisson count observations is concave—however, the result therein showed only a necessary condition for concavity of the expected joint log-likelihood optimised in the M-step.

[3]Our method readily generalises to models in which each dimension $i$ has different nonlinearities $f_i$ and $V_i$.

[4]Again, simply applying SSID to the log of the observed counts does not work as most counts are 0.

# References

[1] R. E. Kalman and R. S. Bucy. New results in linear filtering and prediction theory. *Trans. Am. Soc. Mech. Eng., Series D, Journal of Basic Engineering*, 83:95–108, 1961.

[2] Z. Ghahramani and G. E. Hinton. Parameter estimation for linear dynamical systems. *University of Toronto Technical Report*, 6(CRG-TR-96-2), 1996.

[3] P. V. Overschee and B. D. Moor. N4sid: Subspace algorithms for the identification of combined deterministic-stochastic systems. *Automatica*, 30(1):75–93, 1994.

[4] T. Katayama. *Subspace methods for system identification*. Springer Verlag, 2005.

[5] H. Palanthandalam-Madapusi, S. Lacy, J. Hoagg, and D. Bernstein. Subspace-based identification for linear and nonlinear systems. In *Proceedings of the American Control Conference, 2005*, pp. 2320–2334, 2005.

[6] E. N. Brown, R. E. Kass, and P. P. Mitra. Multiple neural spike train data analysis: state-of-the-art and future challenges. *Nat Neurosci*, 7(5):456–61, 2004.

[7] M. M. Churchland, B. M. Yu, M. Sahani, and K. V. Shenoy. Techniques for extracting single-trial activity patterns from large-scale neural recordings. *Curr Opin Neurobiol*, 17(5):609–618, 2007.

[8] P. McCulloch and J. Nelder. Generalized linear models. *Chapman and Hall, London*, 1989.

[9] K. Yuan and M. Niranjan. Estimating a state-space model from point process observations: a note on convergence. *Neural Comput*, 22(8):1993–2001, 2010.

[10] B. L. Ho and R. E. Kalman. Effective construction of linear state-variable models from input/output functions. *Regelungstechnik*, 14(12):545–548, 1966.

[11] J. Møller, A. Syversveen, and R. Waagepetersen. Log gaussian cox processes. *Scand J Stat*, 25(3):451–482, 1998.

[12] V. Lawhern, W. Wu, N. Hatsopoulos, and L. Paninski. Population decoding of motor cortical activity using a generalized linear model with hidden states. *J Neurosci Methods*, 189(2):267–280, 2010.

[13] A. Z. Mangion, K. Yuan, V. Kadirkamanathan, M. Niranjan, and G. Sanguinetti. Online variational inference for state-space models with point-process observations. *Neural Comput*, 23(8):1967–1999, 2011.

[14] M. Vidne, Y. Ahmadian, J. Shlens, J. Pillow, J. Kulkarni, A. Litke, E. Chichilnisky, E. Simoncelli, and L. Paninski. Modeling the impact of common noise inputs on the network activity of retinal ganglion cells. *J Comput Neurosci*, 2011.

[15] J. H. Macke, L. Büsing, J. P. Cunningham, B. M. Yu, K. V. Shenoy, and M. Sahani. Empirical models of spiking in neural populations. In *Advances in Neural Information Processing Systems*, vol. 24. Curran Associates, Inc., 2012.

[16] J. Kulkarni and L. Paninski. Common-input models for multiple neural spike-train data. *Network*, 18(4):375–407, 2007.

[17] L. Paninski. Maximum likelihood estimation of cascade point-process neural encoding models. *Network*, 15(4):243–262, 2004.

[18] R. E. Turner and M. Sahani. Two problems with variational expectation maximisation for time-series models. In D. Barber, A. T. Cemgil, and S. Chiappa, eds., *Inference and Learning in Dynamic Models*. Cambridge University Press, 2011.

[19] M. Krumin and S. Shoham. Generation of Spike Trains with Controlled Auto-and Cross-Correlation Functions. *Neural Comput*, pp. 1–23, 2009.

[20] J. Macke, P. Berens, A. Ecker, A. Tolias, and M. Bethge. Generating spike trains with specified correlation coefficients. *Neural Comput*, 21(2):397–423, 2009.

[21] B. M. Yu, J. P. Cunningham, G. Santhanam, S. I. Ryu, K. V. Shenoy, and M. Sahani. Gaussian-process factor analysis for low-dimensional single-trial analysis of neural population activity. *J Neurophysiol*, 102(1):614–635, 2009.

[22] M. M. Churchland, B. M. Yu, S. Ryu, G. Santhanam, and K. V. Shenoy. Neural variability in premotor cortex provides a signature of motor preparation. *J Neurosci*, 26(14):3697–3712, 2006.

[23] L. Paninski, Y. Ahmadian, D. Ferreira, S. Koyama, K. Rahnama Rad, M. Vidne, J. Vogelstein, and W. Wu. A new look at state-space models for neural data. *J Comput Neurosci*, 29:107–126, 2010.

[24] K. Yuan, M. Girolami, and M. Niranjan. Markov chain monte carlo methods for state-space models with point process observations. *Neural Comput*, 24(6):1462–1486, 2012.

